# Minimizing Statistical Bias with Queries

**David A. Cohn**
Adaptive Systems Group
Harlequin, Inc.
One Cambridge Center
Cambridge, MA 02142
cohn@harlequin.com

## Abstract

I describe a querying criterion that attempts to minimize the error of a learner by minimizing its estimated squared bias. I describe experiments with locally-weighted regression on two simple problems, and observe that this "bias-only" approach outperforms the more common "variance-only" exploration approach, even in the presence of noise.

## 1 INTRODUCTION

In recent years, there has been an explosion of interest in "active" machine learning systems. These are learning systems that make queries, or perform experiments to gather data that are expected to maximize performance. When compared with "passive" learning systems, which accept given, or randomly drawn data, active learners have demonstrated significant decreases in the amount of data required to achieve equivalent performance. In industrial applications, where each experiment may take days to perform and cost thousands of dollars, a method for optimally selecting these points would offer enormous savings in time and money.

An active learning system will typically attempt to select data that will minimize its predictive error. This error can be decomposed into bias and variance terms. Most research in selecting optimal actions or queries has assumed that the learner is approximately unbiased, and that to minimize learner error, variance is the only thing to minimize (e.g. Fedorov [1972], MacKay [1992], Cohn [1996], Cohn et al., [1996], Paass [1995]). In practice, however, there are very few problems for which we have unbiased learners. Frequently, bias constitutes a large portion of a learner's error; if the learner is deterministic and the data are noise-free, then bias is the *only* source of error. Note that the bias term here is a *statistical* bias, distinct from the *inductive* bias discussed in some machine learning research [Dietterich and Kong, 1995].

In this paper I describe an algorithm which selects actions/queries designed to minimize the bias of a locally weighted regression-based learner. Empirically, "variance-minimizing" strategies which ignore bias seem to perform well, even in cases where, strictly speaking, there is no variance to minimize. In the tasks considered in this paper, the bias-minimizing strategy consistently outperforms variance minimization, even in the presence of noise.

## 1.1  BIAS AND VARIANCE

Let us begin by defining $P(x, y)$ to be the unknown joint distribution over $x$ and $y$, and $P(x)$ to be the known marginal distribution of $x$ (commonly called the *input distribution*). We denote the learner's output on input $x$, given training set $\mathcal{D}$ as $\hat{y}(x; \mathcal{D})$. We can then write the expected error of the learner as

$$\int_x E\left[\left(\hat{y}(x; \mathcal{D}) - y(x)\right)^2 | x\right] P(x) dx, \tag{1}$$

where $E[\cdot]$ denotes the expectation over $P$ and over training sets $\mathcal{D}$. The expectation inside the integral may be decomposed as follows (Geman et al., 1992):

$$
\begin{aligned}
E\left[\left(\hat{y}(x; \mathcal{D}) - y(x)\right)^2 | x\right] =\ & E\left[\left(y(x) - E[y|x]\right)^2\right] \\
& + \left(E_{\mathcal{D}}\left[y(x; \mathcal{D})\right] - E[y|x]\right)^2 \\
& + E_{\mathcal{D}}\left[\left(\hat{y}(x; \mathcal{D}) - E_{\mathcal{D}}[\hat{y}(x; \mathcal{D})]\right)^2\right]
\end{aligned}
\tag{2}
$$

where $E_{\mathcal{D}}[\cdot]$ denotes the expectation over training sets. The first term in Equation 2 is the variance of $y$ given $x$ – it is the *noise* in the distribution, and does not depend on our learner or how the training data are chosen. The second term is the learner's *squared bias*, and the third is its *variance*; these last two terms comprise the expected squared error of the learner with respect to the regression function $E[y|x]$.

Most research in active learning assumes that the second term of Equation 2 is approximately zero, that is, that the learner is *unbiased*. If this is the case, then one may concentrate on selecting data so as to minimize the variance of the learner. Although this "all-variance" approach is optimal when the learner is unbiased, truly unbiased learners are rare. Even when the learner's representation class is able to match the target function exactly, bias is generally introduced by the learning algorithm and learning parameters. From the Bayesian perspective, a learner is only unbiased if its priors are *exactly* correct.

The optimal choice of query would, of course, minimize *both* bias and variance, but I leave that for future work. For the purposes of this paper, I will only be concerned with selecting queries that are expected to minimize learner bias. This approach is justified in cases where noise is believed to be only a small component of the learner's error. If the learner is deterministic and there is no noise, then strictly speaking, there *is* no error due to variance — all the error must be due to learner bias. In cases with non-determinism or noise, all-bias minimization, like all-variance minimization, becomes an approximation of the optimal approach.

The learning model discussed in this paper is a form of locally weighted regression (LWR) [Cleveland et al., 1988], which has been used in difficult machine learning tasks, notably the "robot juggler" of Schaal and Atkeson [1994]. Previous work [Cohn et al., 1996] discussed all-variance query selection for LWR; in the remainder of this paper, I describe a method for performing all-bias query selection. Section 2 describes the criterion that must be optimized for all-bias query selection. Section 3 describes the locally weighted regression learner used in this paper and describes

how the all-bias criterion may be computed for it. Section 4 describes the results of experiments using this criterion on several simple domains. Directions for future work are discussed in Section 5.

## 2  ALL-BIAS QUERY SELECTION

Let us assume for the moment that we have a source of noise-free examples $(x_i, y_i)$ and a deterministic learner which, given input $x$, outputs estimate $\hat{y}(x)$.[1] Let us also assume that we have an accurate estimate of the bias of $\hat{y}$ which can be used to estimate the true function $y(x) = \hat{y}(x) - bias(x)$. We will break these rather strong assumptions of noise-free examples and accurate bias estimates in Section 4, but they are useful for deriving the theoretical approach described below.

Given an accurate bias estimate, we must force the biased estimator into the best approximation of $y(x)$ with the fewest number of examples. This, in effect, transforms the query selection problem into an example filter problem similar to that studied by Plutowski and White [1993] for neural networks. Below, I derive this criterion for estimating the change in error at $x$ given a new queried example at $\tilde{x}$.

Since we have (temporarily) assumed a deterministic learner and noise-free data, the expected error in Equation 2 simplifies to:

$$E\left[(\hat{y}(x;\mathcal{D}) - y(x))^2 \,|x, \mathcal{D}\right] \;=\; (\hat{y}(x;\mathcal{D}) - y(x))^2 \qquad (3)$$

We want to select a new $\tilde{x}$ such that when we add $(\tilde{x}, \tilde{y})$, the resulting squared bias is minimized:

$$(\hat{y}' - y)^2 \equiv (\hat{y}(x;\mathcal{D} \cup (\tilde{x}, \tilde{y})) - y(x))^2 . \qquad (4)$$

I will, for the remainder of the paper, use the "′" to indicate estimates based on the initial training set plus the additional example $(\tilde{x}, \tilde{y})$. To minimize Expression 4, we need to compute how a query at $\tilde{x}$ will change the learner's bias at $x$. If we assume that we know the input distribution,[2] then we can integrate this change over the entire domain (using Monte Carlo procedures) to estimate the resulting average change, and select a $\tilde{x}$ such that the expected squared bias is minimized. Defining $bias \equiv \hat{y} - y$ and $\Delta\hat{y} \equiv \hat{y}' - \hat{y}$, we can write the new squared bias as:

$$\begin{aligned} bias'^2 &= (\hat{y}' - y)^2 = (\hat{y} + \Delta\hat{y} - y)^2 \\ &= \Delta\hat{y}^2 + 2\Delta\hat{y} \cdot bias + bias^2 \end{aligned} \qquad (5)$$

Note that since *bias* as defined here is independent of $\tilde{x}$, minimizing the bias is equivalent to minimizing $\Delta\hat{y}^2 + 2\Delta\hat{y} \cdot bias$.

The estimate of *bias'* tells us how much our bias will change for a given $\tilde{x}$. We may optimize this value over $\tilde{x}$ in one of a number of ways. In low dimensional spaces, it is often sufficient to consider a set of "candidate" $\tilde{x}$ and select the one promising the smallest resulting error. In higher dimensional spaces, it is often more efficient to search for an optimal $\tilde{x}$ with a response surface technique [Box and Draper, 1987], or hillclimb on $\partial bias'^2 / \partial \tilde{x}$.

Estimates of *bias* and $\Delta\hat{y}$ depend on the specific learning model being used. In Section 3, I describe a locally weighted regression model, and show how differentiable estimates of *bias* and $\Delta\hat{y}$ may be computed for it.

## 2.1  AN ASIDE: WHY NOT JUST USE $\hat{y} - \widehat{bias}$?

If we have an accurate bias estimate, it is reasonable to ask why we do not simply use the corrected $\hat{y} - \widehat{bias}$ as our predictor. The answer has two parts, the first of which is that for most learners, there are no perfect bias estimators — they introduce their own bias and variance, which must be addressed in data selection.

Second, we *can* define a composite learner $\hat{y}_c \equiv \hat{y} - \widehat{bias}$. Given a random training sample then, we would expect $\hat{y}_c$ to outperform $\hat{y}$. However, there is no obvious way to select data for this composite learner other than selecting to maximize the performance of its two components. In our case, the second component (the bias estimate) is non-analytic, which leaves us selecting data so as to maximize the performance of the first component (the uncorrected estimator). We are now back to our original problem: we can select data so as to minimize either the bias or variance of the uncorrected LWR-based learner. Since the purpose of the correction is to give an unbiased estimator, intuition suggests that variance minimization would be the more sensible route in this case. Empirically, this approach does not appear to yield any benefit over uncorrected variance minimization (see Figure 1).

## 3  LOCALLY WEIGHTED REGRESSION

The type of learner I consider here is a form of locally weighted regression (LWR) that is a slight variation on the LOESS model of Cleveland et al. [1988] (see Cohn et al., [1996] for details). The LOESS model performs a linear regression on points in the data set, weighted by a kernel centered at $x$. The kernel shape is a design parameter: the original LOESS model uses a "tricubic" kernel; in my experiments I use the more common Gaussian

$$h_i(x) \equiv h(x - x_i) = \exp(-k(x - x_i)^2),$$

where $k$ is a smoothing parameter. For brevity, I will drop the argument $x$ for $h_i(x)$, and define $n = \sum_i h_i$. We can then write the weighted means and covariances as:

$$\mu_x = \sum_i h_i \frac{x_i}{n}, \quad \sigma_x^2 = \sum_i h_i \frac{(x_i - x)^2}{n}, \quad \sigma_{y|x}^2 = \sigma_y^2 - \frac{\sigma_{xy}^2}{\sigma_x^2},$$

$$\mu_y = \sum_i h_i \frac{y_i}{n}, \quad \sigma_y^2 = \sum_i h_i \frac{(y_i - \mu_y)^2}{n}, \quad \sigma_{xy} = \sum_i h_i \frac{(x_i - x)(y_i - \mu_y)}{n}.$$

We use these means and covariances to produce an estimate $\hat{y}$ at the $x$ around which the kernel is centered, with a confidence term in the form of a variance estimate:

$$\hat{y} = \mu_y + \frac{\sigma_{xy}}{\sigma_x^2}(x - \mu_x) \quad \sigma_{\hat{y}}^2 = \sigma_{y|x}^2 \frac{\sum_i h_i^2}{n^2} \left[ 1 + \frac{(x - \mu_x)(x_i - \mu_x)}{\sigma_x^2} \right]^2.$$

In all the experiments discussed in this paper, the smoothing parameter $k$ was set so as to minimize $\sigma_{\hat{y}}^2$.

The low cost of incorporating new training examples makes this form of locally weighted regression appealing for learning systems which must operate in real time, or with time-varying target functions (e.g. [Schaal and Atkeson 1994]).

## 3.1 COMPUTING $\Delta\hat{y}$ FOR LWR

If we know what new point $(\tilde{x}, \tilde{y})$ we're going to add, computing $\Delta\hat{y}$ for LWR is straightforward. Defining $\tilde{h}$ as the weight given to $\tilde{x}$, and $\tilde{n}$ as $n + \tilde{h}$ we can write

$$
\begin{aligned}
\Delta\hat{y} &= \hat{y}' - \hat{y} = \mu_y' + \frac{\sigma_{xy}'}{\sigma_x'^2}(x - \mu_x') - \mu_y - \frac{\sigma_{xy}}{\sigma_x^2}(x - \mu_x) \\
&= \tilde{h}\frac{(\tilde{y} - \mu_y)}{\tilde{n}} - \frac{\sigma_{xy}}{\sigma_x^2}(x - \mu_x) + \left(x - \frac{n\mu_x}{\tilde{n}} - \frac{\tilde{h}\tilde{x}}{\tilde{n}}\right) \cdot \frac{\tilde{n}\sigma_{xy} + \tilde{h} \cdot (\tilde{x} - \mu_x)(\tilde{y} - \mu_y)}{\tilde{n}\sigma_x^2 + \tilde{h} \cdot (\tilde{x} - \mu_x)^2}
\end{aligned}
$$

Note that computing $\Delta\hat{y}$ requires us to know both the $\tilde{x}$ and $\tilde{y}$ of the new point. In practice, we only know $\tilde{x}$. If we assume, however, that we can estimate the learner's bias at any $x$, then we can also estimate the unknown value $\tilde{y} \approx \hat{y}(\tilde{x}) - bias(\tilde{x})$. Below, I consider how to compute the bias estimate.

## 3.2 ESTIMATING BIAS FOR LWR

The most common technique for estimating bias is cross-validation. Standard cross-validation however, only gives estimates of the bias at our specific training points, which are usually combined to form an average bias estimate. This is sufficient if one assumes that the training distribution is representative of the test distribution (which it isn't in query learning) and if one is content to just estimate the bias where one already has training data (which we can't be).

In the query selection problem, we must be able to estimate the bias at all possible $x$. Box and Draper [1987] suggest fitting a higher order model and measuring the difference. For the experiments described in this paper, this method yielded poor results; two other bias-estimation techniques, however, performed very well.

One method of estimating bias is by bootstrapping the residuals of the training points. One produces a "bootstrap sample" of the learner's residuals on the training data, and adds them to the original predictions to create a synthetic training set. By averaging predictions over a number of bootstrapped training sets and comparing the average prediction with that of the original predictor, one arrives at a first-order bootstrap estimate of the predictor's bias [Connor 1993; Efron and Tibshirani, 1993]. It is known that this estimate is itself biased towards zero; a standard heuristic is to divide the estimate by 0.632 [Efron, 1983].

Another method of estimating bias of a learner is by fitting its own cross-validated residuals. We first compute the cross-validated residuals on the training examples. These produce estimates of the learner's bias at each of the training points. We can then use these residuals as training examples for another learner (again LWR) to produce estimates of what the cross-validated error would be in places where we don't have training data.

## 4 EMPIRICAL RESULTS

In the previous two sections, I have explained how having an estimate of $\Delta\hat{y}$ and *bias* for a learner allows one to compute the learner's change in bias given a new query, and have shown how these estimates may be computed for a learner that uses locally weighted regression. Here, I apply these results to two simple problems and demonstrate that they may actually be used to select queries that minimize the statistical bias (and the error) of the learner. The problems involve learning the kinematics of a planar two-jointed robot arm: given the shoulder and elbow joint angles, the learner must predict the tip position.

## 4.1 BIAS ESTIMATES

I tested the accuracy of the two bias estimators by observing their correlations on 64 reference inputs, given 100 random training examples from the planar arm problem. The bias estimates had a correlation with actual biases of 0.852 for the bootstrap method, and 0.871 for the cross-validation method.

## 4.2 BIAS MINIMIZATION

I ran two sets of experiments using the bias-minimizing criterion in conjunction with the bias estimation technique of the previous section on the planar arm problem. The bias minimization criterion was used as follows: At each time step, the learner was given a set of 64 randomly chosen candidate queries and 64 uniformly chosen reference points. It evaluated $E'(x)$ for each reference point given each candidate point and selected for its next query the candidate point with the smallest average $E'(x)$ over the reference points. I compared the bias-minimizing strategy (using the cross-validation and bootstrap estimation techniques) against random sampling and the variance-minimizing strategy discussed in Cohn et al. [1996]. On a Sparc 10, with $m$ training examples, the average evaluation times per candidate per reference point were $58 + 0.16m$ $\mu$seconds for the variance criterion, $65 + 0.53m$ $\mu$seconds for the cross-validation-based bias criterion, and $83 + 3.7m$ $\mu$seconds for the bootstrap-based bias criterion (with 20x resampling).

To test whether the bias-only assumption was robust against the presence of noise, 1% Gaussian noise was added to the input values of the training data in all experiments. This simulates noisy position effectors on the arm, and results in non-Gaussian noise in the output coordinate system.

In the first series of experiments, the candidate shoulder and elbow joint angles were drawn uniformly over $(U[0, 2\pi], U[0, \pi])$. In unconstrained domains like this, random sampling is a fairly good default strategy. The bias minimization strategies still significantly outperform both random sampling and the variance minimizing strategy in these experiments (see Figure 1).

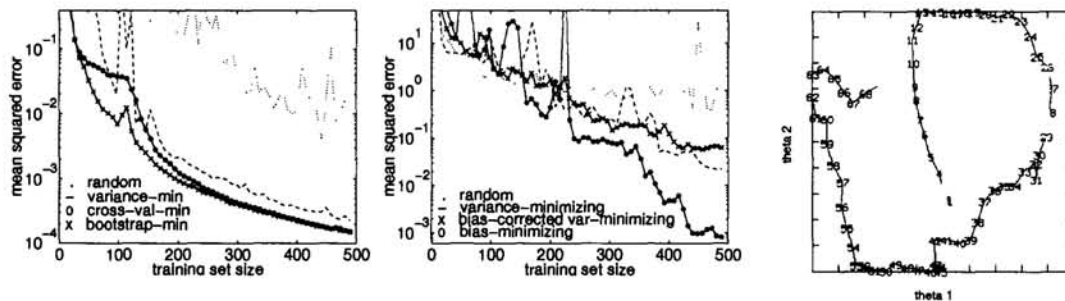

Figure 1: (left) MSE as a function of number of noisy training examples for the unconstrained arm problem. Errors are averaged over 10 runs for the bootstrap method and 15 runs for all others. One run with the cross-validation-based method was excluded when $k$ failed to converge to a reasonable value. (center) MSE as a function of number of noisy training examples for the constrained arm problem. The bias correction strategy discussed in Section 2.1 does no better than the uncorrected variance-minimizing strategy, and much worse than the bias-minimization strategy. (right) Sample exploration trajectory in joint-space for the constrained arm problem, explored according to the bias minimizing criterion.

In the second series of experiments, candidates were drawn uniformly from a region

local to the previously selected query: $(\hat{\theta}_1 \pm 0.2\pi, \hat{\theta}_2 \pm 0.1\pi)$. This corresponds to restricting the arm to local motions. In a constrained problem such as this, random sampling is a poor strategy; both the bias and variance-reducing strategies outperform it at least an order of magnitude. Further, the bias-minimization strategy outperforms variance minimization by a large margin (Figure 1). Figure 1 also shows an exploration trajectory produced by pursuing the bias-minimizing criterion. It is noteworthy that, although the implementation in this case was a greedy (one-step) minimization, the trajectory results in globally good exploration.

## 5 DISCUSSION

I have argued in this paper that, in many situations, selecting queries to minimize learner bias is an appropriate and effective strategy for active learning. I have given empirical evidence that, with a LWR-based learner and the examples considered here, the strategy is effective even in the presence of noise.

Beyond minimizing either bias or variance, an important next step is to explicitly minimize them together. The bootstrap-based estimate should facilitate this, as it produces a complementary variance estimate with little additional computation. By optimizing over both criteria simultaneously, we expect to derive a criterion that that, in terms of statistics, is truly optimal for selecting queries.

## Footnotes

[1]For clarity, I will drop the argument $x$ except where required for disambiguation. I will also denote only the univariate case; the results apply in higher dimensions as well.

[2]This assumption is contrary to the assumption normally made in some forms of learning, e.g. PAC-learning, but it is appropriate in many domains.

## REFERENCES

**Box, G., & Draper, N.** (1987). *Empirical model-building and response surfaces*, Wiley, New York.

**Cleveland, W., Devlin, S., & Grosse, E.** (1988). Regression by local fitting. *Journal of Econometrics*, **37**, 87–114.

**Cohn, D.** (1996) Neural network exploration using optimal experiment design. *Neural Networks*, 9(6):1071–1083.

**Cohn, D., Ghahramani, Z., & Jordan, M.** (1996). Active learning with statistical models. *Journal of Artificial Inteligence Research* 4:129–145.

**Connor, J.** (1993). Bootstrap Methods in Neural Network Time Series Prediction. In J. Alspector et al., eds., *Proc. of the Int. Workshop on Applications of Neural Networks to Telecommunications*, Lawrence Erlbaum, Hillsdale, N.J.

**Dietterich, T., & Kong, E.** (1995). Error-correcting output coding corrects bias and variance. In S. Prieditis and S. Russell, eds., *Proceedings of the 12th International Conference on Machine Learning*.

**Efron, B.** (1983) Estimating the error rate of a prediction rule: some improvements on cross-validation. *J. Amer. Statist. Assoc.* **78**:316–331.

**Efron, B. & Tibshirani, R.** (1993). *An introduction to the bootstrap*. Chapman & Hall, New York.

**Fedorov, V.** (1972). *Theory of Optimal Experiments*. Academic Press, New York.

**Geman, S., Bienenstock, E., & Doursat, R.** (1992). Neural networks and the bias/variance dilemma. *Neural Computation*, **4**, 1–58.

**MacKay, D.** (1992). Information-based objective functions for active data selection, *Neural Computation*, **4**, 590–604.

**Paass, G., and Kindermann, J.** (1994). Bayesian Query Construction for Neural Network Models. In G. Tesauro et al., eds., *Advances in Neural Information Processing Systems 7*, MIT Press.

**Plutowski, M., & White, H.** (1993). Selecting concise training sets from clean data. *IEEE Transactions on Neural Networks*, **4**, 305–318.

**Schaal, S. & Atkeson, C.** (1994). Robot Juggling: An Implementation of Memory-based Learning. *Control Systems* **14**, 57–71.